# Anatomical origin and computational role of diversity in the response properties of cortical neurons

**Kalanit Grill Spector†** **Shimon Edelman†** **Rafael Malach‡**

Depts of †Applied Mathematics and Computer Science and ‡Neurobiology
The Weizmann Institute of Science
Rehovot 76100, Israel
{kalanit,edelman,malach}@wisdom.weizmann.ac.il

## Abstract

The maximization of diversity of neuronal response properties has been recently suggested as an organizing principle for the formation of such prominent features of the functional architecture of the brain as the cortical columns and the associated patchy projection patterns (Malach, 1994). We show that (1) maximal diversity is attained when the ratio of dendritic and axonal arbor sizes is equal to one, as found in many cortical areas and across species (Lund et al., 1993; Malach, 1994), and (2) that maximization of diversity leads to better performance in systems of receptive fields implementing steerable/shiftable filters, and in matching spatially distributed signals, a problem that arises in many high-level visual tasks.

## 1 Anatomical substrate for sampling diversity

A fundamental feature of cortical architecture is its columnar organization, manifested in the tendency of neurons with similar properties to be organized in columns that run perpendicular to the cortical surface. This organization of the cortex was initially discovered by physiological experiments (Mouncastle, 1957; Hubel and Wiesel, 1962), and subsequently confirmed with the demonstration of histologically defined columns. Tracing experiments have shown that axonal projections throughout the cerebral cortex tend to be organized in vertically aligned clusters or patches. In particular, intrinsic horizontal connections linking neighboring cortical sites, which may extend up to $2 - 3$ $mm$, have a striking tendency to arborize selectively in preferred sites, forming distinct axonal patches $200 - 300$ $\mu m$ in diameter.

Recently, it has been observed that the size of these patches matches closely the average diameter of individual dendritic arbors of upper-layer pyramidal cells

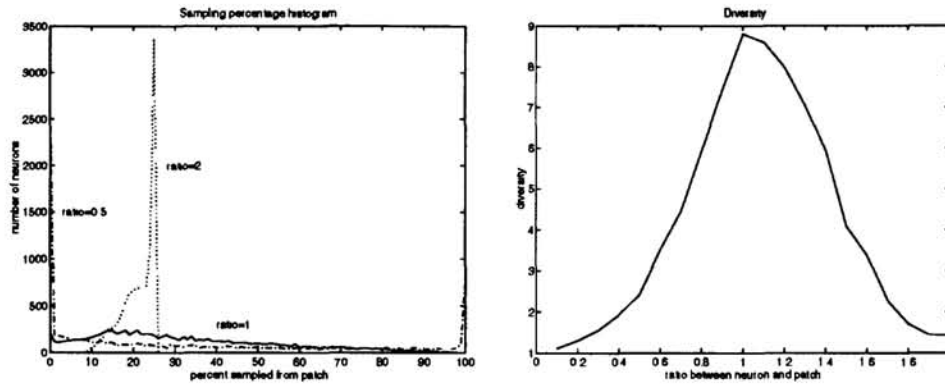

Figure 1: *Left:* histograms of the percentage of patch-originated input to the neurons, plotted for three values of the ratio $r$ between the dendritic arbor and the patch diameter (0.5, 1.0, 2.0). The flattest histogram is obtained for $r = 1.0$ *Right:* the diversity of neuronal properties (as defined in section 1) vs. $r$. The maximum is attained for $r = 1.0$, a value compatible with the anatomical data.

(see Malach, 1994, for a review). Determining the functional significance of this correlation, which is a fundamental property that holds throughout various cortical areas and across species (Lund et al., 1993), may shed light on the general principles of operation of the cortical architecture. One such driving principle may be the *maximization of diversity* of response properties in the neuronal population (Malach, 1994). According to this hypothesis, matching the sizes of the axonal patches and the dendritic arbors causes neighboring neurons to develop slightly different functional selectivity profiles, resulting in an even spread of response preferences across the cortical population, and in an improvement of the brain's ability to process the variety of stimuli likely to be encountered in the environment.[1]

To test the effect of the ratio between axonal patch and dendritic arbor size on the diversity of the neuronal population, we conducted computer simulations based on anatomical data concerning patchy projections (Rockland and Lund, 1982; Lund et al., 1993; Malach, 1992; Malach et al., 1993). The patches were modeled by disks, placed at regular intervals of twice the patch diameter, as revealed by anatomical labeling. Dendritic arbors were also modeled by disks, whose radii were manipulated in different simulations. The arbors were placed randomly over the axonal patches, at a density of 10,000 neurons per patch. We then calculated the amount of patch-related information sampled by each neuron, defined to be proportional to the area of overlap of the dendritic tree and the patch. The results of the calculations for three

values of the ratio of patch and arbor diameters appear in Figure 1.

The presence of two peaks in the histogram obtained with the arbor/patch ratio $r = 0.5$ indicates that two dominant groups are formed in the population, the first receiving most of its input from the patch, and the second – from the inter-patch sources. A value of $r = 2.0$, for which the dendritic arbors are larger than the axonal patch size, yields near uniformity of sampling properties, with most of the neurons receiving mostly patch-originated input, as apparent from the single large peak in the histogram. To quantify the notion of diversity, we defined it as *diversity* $\sim < |\frac{dn}{dp}| >^{-1}$, where $n(p)$ is the number of neurons that receive $p$ percent of their inputs from the patch, and $< \cdot >$ denotes average over $p$. Figure 1, right, shows that diversity is maximized when the size of the dendritic arbors matches that of the axonal patches, in accordance with the anatomical data. This result confirms the diversity maximization hypothesis stated in (Malach, 1994).

# 2   Orientation tuning: a functional manifestation of sampling diversity

The orientation columns in V1 are perhaps the best-known example of functional architecture found in the cortex (Hubel and Wiesel, 1962). Cortical maps obtained by optical imaging (Grinvald et al., 1986) reveal that orientation columns are patchy rather then slab-like: domains corresponding to a single orientation appear as a mosaic of round patches, which tend to form pinwheel-like structures. Incremental changes in the orientation of the stimulus lead to smooth shifts in the position of these domains. We hypothesized that this smooth variation in orientation selectivity found in V1 originates in patchy projections, combined with diversity in the response properties of neurons sampling from these projections. The simulations described in the rest of this section substantiate this hypothesis.

**Computer simulations.**   The goal of the simulations was to demonstrate that a limited number of discretely tuned elements can give rise to a continuum of responses. We did not try to explain how the original set of discrete orientations can be formed by projections from the LGN to the striate cortex; several models for this step can be found in the literature (Hubel and Wiesel, 1962; Vidyasagar, 1985).[2] In setting the size of the original discrete orientation columns we followed the notion of a point image (MacIlwain, 1986), defined as the minimal cortical separation of cells with non-overlapping RFs. Each column was tuned to a specific angle, and located at an approximately constant distance from another column with the same orientation tuning (we allowed some scatter in the location of the RFs). The RFs of adjacent units with the same orientation preference were overlapping, and the amount of overlap

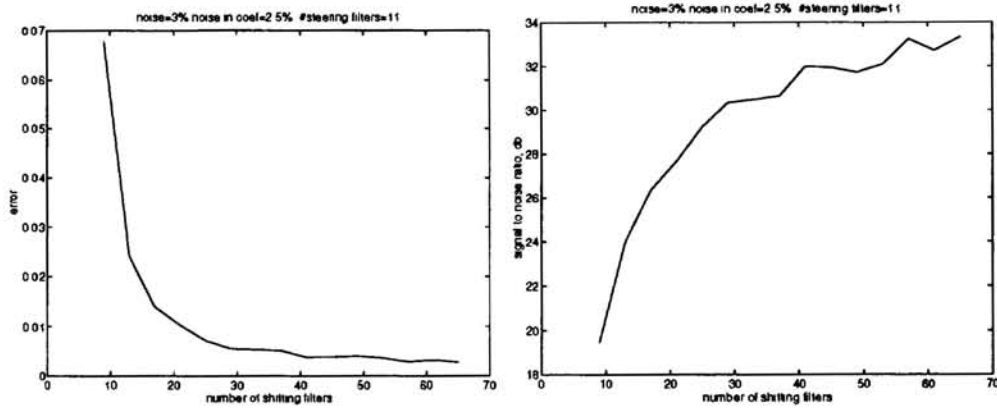

Figure 2: The effects of (independent) noise in the basis RFs and in the steering/shifting coefficients. *Left:* the approximation error vs. the number of basis RFs used in the linear combination. *Right:* the signal to noise ratio vs. the number of basis RFs. The SNR values were calculated as $10 \log_{10}$ (*signal energy/noise energy*). Adding RFs to the basis increases the accuracy of the resultant interpolated RF.

was determined by the number of RFs incorporated into the network. The preferred orientations were equally spaced at angles between 0 and $\pi$. The RFs used in the simulations were modeled by a product of a 2D Gaussian $G_1$, centered at $\vec{r_j}$, with orientation selectivity $G_2$, and optimal angle $\theta_i$: $G(\vec{r}, \vec{r_j}, \theta, \theta_i) = G_1(\vec{r}, \vec{r_j})G_2(\theta, \theta_i)$.

According to the recent results on shiftable/steerable filters (Simoncelli et al., 1992), a RF located at $\vec{r_0}$ and tuned to the orientation $\phi_0$ can be obtained by a linear combination of basis RFs, as follows:

$$
\begin{aligned}
G(\vec{r}, \vec{r_0}, \theta, \phi_0) &= \sum_{j=0}^{M-1} \sum_{i=0}^{N-1} b_j(\vec{r_0}) k_i(\phi_0) G(\vec{r}, \vec{r_j}, \theta, \theta_i) \\
&= \sum_{j=0}^{M-1} b_j(\vec{r_0}) G_1(\vec{r}, \vec{r_j}) \sum_{i=0}^{N-1} k_i(\phi_0) G_2(\theta, \theta_i)
\end{aligned}
\tag{1}
$$

From equation 1 it is clear that the linear combination is equivalent to an outer product of the shifted and the steered RFs, with $\{k_i(\phi_0)\}_{i=0}^{N-1}$ and $\{b_j(\vec{r_0})\}_{j=0}^{M-1}$ denoting the steering and shifting coefficients, respectively. Because orientation and localization are independent parameters, the steering coefficients can be calculated separately from the shifting coefficients. The number of steering coefficients depends on the polar Fourier bandwidth of the basis RF, while the number of steering filters is inversely proportional to the basis RF size (Grill-Spector et al., 1995). In the presence of noise this minimal basis has to be extended (see Figure 2). The results of the simulation for several RF sizes are shown in Figure 3, left. As expected, the number of basis RFs required to approximate a desired RF is inversely proportional to the

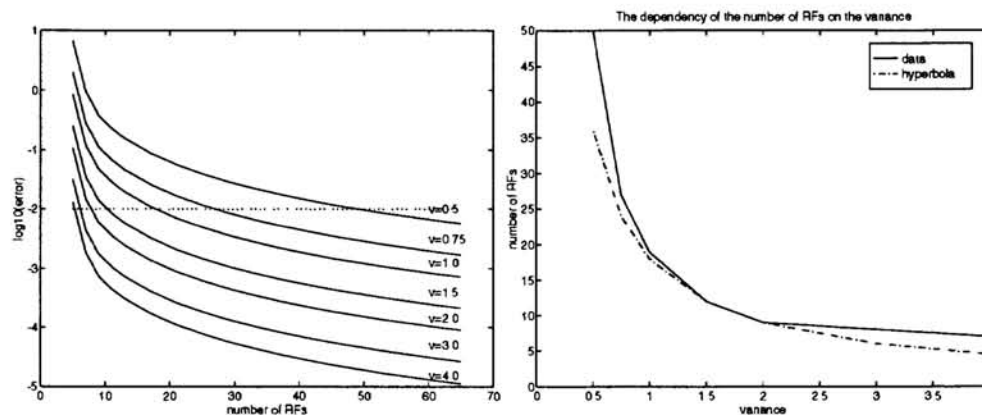

Figure 3: *Left:* error of the steering/shifting approximation for several basis RF sizes. *Right:* the number of basis RFs required to achieve a given error for different sizes of the basis RFs. The dashed line is the hyperbola $num\ RFs \times size = const$.

size of the basis RFs (Figure 3, right).

**Steerability and biological considerations.** The anatomical finding that the columnar "borders" are freely crossed by dendritic and axonal arbors (Malach, 1992), and the mathematical properties of shiftable/steerable filters outlined above suggest that the columnar architecture in V1 provides a basis for creating a continuum of RF properties, rather that being a form of organizing RFs in discrete bins. Computationally, this may be possible if the input to neurons is a linear combination of outputs of several RFs, as in equation 1. The anatomical basis for this computation may be provided by intrinsic cortical connections. It is known that long-range ($\sim 1mm$) connections tend to link cells with like orientation preference, while the short-range ($\sim 400\ \mu m$) connections are made to cells of diverse orientation preferences (Malach et al., 1993). We suggest that the former provide the inputs necessary to shift the position of the desired RF, while the latter participate in steering the RF to an arbitrary angle (see Grill-Spector et al., 1995, for details).

# 3  Matching with patchy connections

Many visual tasks require matching between images taken at different points in space (as in binocular stereopsis) or time (as in motion processing). The first and foremost problem faced by a biological system in solving these tasks is that the images to be compared are not represented as such anywhere in the system: instead of images, there are patterns of activities of neurons, with RFs that are overlapping, are not located on a precise grid, and are subject to mixing by patchy projections in each successive stage of processing. In this section, we show that a system composed of scattered RFs with smooth and overlapping tuning functions can, as a matter of

fact, perform matching precisely by allowing patchy connections between domains. Moreover, the weights that must be given to the various inputs that feed a RF carrying out the match are identical to the coefficients that would be generated by a learning algorithm required to capture a certain well-defined input-output relationship from pairs of examples.

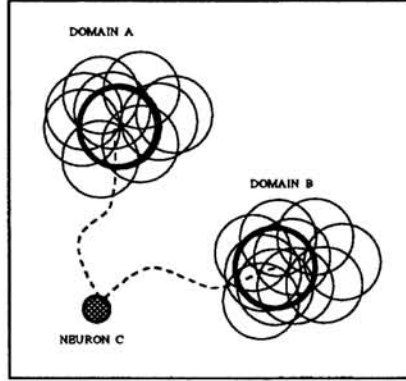

Figure 4: Unit C receives patchy input from areas A and B which contain receptors with overlapping RFs.

Consider a unit C, sampling two domains A and B through a Gaussian-profile dendritic patch equal in size to that of the axonal arbor of cells feeding A and B (Figure 4). The task faced by unit C is to determine the degree to which the activity patterns in domains A and B match. Let $\phi_{jp}$ be the response of the $j$'th unit in A to an input $\vec{x_p}$:

$$\phi_{jp} = \exp\{\frac{-(\vec{x_p} - \vec{x_j})^2}{2\sigma^2}\} \tag{2}$$

where $\vec{x_j}$ be the optimal pattern to which the $j$'th unit is tuned (the response $\theta_{jp}$ of a unit in B is of similar form). If, for example, domains A and B contain orientation selective cells, then $\vec{x_j}$ would be the optimal combination of orientation and location of a bar stimulus. For simplicity we assume that all the RFs are of the same size $\sigma$, that unit C samples the same number of neurons $N$ from both domains, and that the input from each domain to unit C is a linear combination of the responses of the units in each area. The input to C from domain A, with $\vec{x_p}$ presented to the system is then:

$$A_{in} = \sum_{j=1}^{N} a_j \phi_{jp} \tag{3}$$

The problem is to find coefficients $\{a_j\}$ and $\{b_j\}$ such that on a given set of inputs $\{\vec{x}_p\}$ the outputs of domains A and B will match. We define the matching error as follows:

$$E_m = \sum_{p=1}^{P} \left( \sum_{i=1}^{N} a_i \phi_{ip} - \sum_{i=1}^{N} b_i \theta_{ip} \right)^2 \qquad (4)$$

**Proposition 1** *The desired coefficients, minimizing $E_m$, can be generated by an algorithm trained to learn an input/output mapping from a set of examples.*

This proposition can be proved by taking the derivative of $E_m$ with respect to the coefficients (Grill-Spector et al., 1995). Learning here can be carried out by radial basis function (RBF) approximation (Poggio and Girosi, 1990), which is particularly suitable for our purpose, because its basis functions can be regarded as multidimensional Gaussian RFs.

# 4   Summary

Our results show that maximal diversity of neuronal response properties is attained when the ratio of dendritic and axonal arbor sizes is equal to 1, a value found in many cortical areas and across species (Lund et al., 1993; Malach, 1994). Maximization of diversity also leads to better performance in systems of receptive fields implementing steerable/shiftable filters, which may be necessary for generating the seemingly continuous range of orientation selectivity found in V1, and in matching spatially distributed signals. This cortical organization principle may, therefore, have the double advantage of accounting for the formation of the cortical columns and the associated patchy projection patterns, and of explaining how systems of receptive fields can support functions such as the generation of precise response tuning from imprecise distributed inputs, and the matching of distributed signals, a problem that arises in visual tasks such as stereopsis, motion processing, and recognition.

## Footnotes

[1]Necessary conditions for obtaining dendritic sampling diversity are that dendritic arbors cross freely through column borders, and that dendrites which cross column borders sample with equal probability from patch and inter-patch compartments. These assumptions were shown to be valid in (Malach, 1992; Malach, 1994).

[2]In particular, it has been argued (Vidyasagar, 1985) that the receptive fields at the output of the LGN are already broadly tuned for a small number of discrete orientations (possibly just horizontal and vertical), and that at the cortical level the entire spectrum of orientations is generated from the discrete set present in the geniculate projection.

# References

Grill-Spector, K., Edelman, S., and Malach, R. (1995). Anatomical origin and computational role of diversity in the response properties of cortical neurons. In Aertsen, A., editor, *Brain Theory: biological basis and computational theory of vision*. Elsevier. in press.

Grinvald, A., Lieke, T., Frostigand, R., Gilbert, C., and Wiesel, T. (1986). Functional architecture of the cortex as revealed by optical imaging of intrinsic signals. *Nature*, 324:361–364.

Hubel, D. and Wiesel, T. (1962). Receptive fields, binocular interactions and functional architecture in the cat's visual cortex. *Journal of Physiology*, 160:106–154.

Lund, J., Yoshita, S., and Levitt, J. (1993). Comparison of intrinsic connections in different areas of macaque cerebral cortex. *Cerebral Cortex*, 3:148–162.

MacIlwain, J. (1986). Point images in the visual system: new interest in an old idea. *Trends in Neurosciences*, 9:354–358.

Malach, R. (1992). Dendritic sampling across processing streams in monkey striate cortex. *Journal of Comparative Neurobiology*, 315:305–312.

Malach, R. (1994). Cortical columns as devices for maximizing neuronal diversity. *Trends in Neurosciences*, 17:101–104.

Malach, R., Amir, Y., Harel, M., and Grinvald, A. (1993). Relationship between intrinsic connections and functional architecture,revealed by optical imaging and in vivo targeted biocytine injections in primate striate cortex. *Proceedings of the National Academy of Science, USA*, 90:10469–10473.

Mouncastle, V. (1957). Modality and topographic properties of single neurons of cat's somatic sensory cortex. *Journal of Neurophysiology*, 20:408–434.

Poggio, T. and Girosi, F. (1990). Regularization algorithms for learning that are equivalent to multilayer networks. *Science*, 247:978–982.

Rockland, K. and Lund, J. (1982). Widespread periodic intrinsic connections in the tree shrew visual cortex. *Science*, 215:1532–1534.

Simoncelli, E., Freeman, W., Adelson, E., and Heeger, D. (1992). Shiftable multiscale transformations. *IEEE Transactions on Information Theory*, 38:587–607.

Vidyasagar, T. (1985). Geniculate orientation biases as cartesian coordinates for cortical orientation detectors. In *Models for the visual cortex*, pages 390–395. Wiley, New York.